# Clustering appearance and shape by learning jigsaws

**Anitha Kannan, John Winn, Carsten Rother**
Microsoft Research Cambridge
`[ankannan, jwinn, carrot]@microsoft.com`

## Abstract

Patch-based appearance models are used in a wide range of computer vision applications. To learn such models it has previously been necessary to specify a suitable set of patch sizes and shapes by hand. In the jigsaw model presented here, the shape, size and appearance of patches are learned automatically from the repeated structures in a set of training images. By learning such irregularly shaped 'jigsaw pieces', we are able to discover both the shape and the appearance of object parts without supervision. When applied to face images, for example, the learned jigsaw pieces are surprisingly strongly associated with face parts of different shapes and scales such as eyes, noses, eyebrows and cheeks, to name a few. We conclude that learning the shape of the patch not only improves the accuracy of appearance-based part detection but also allows for shape-based part detection. This enables parts of similar appearance but different shapes to be distinguished; for example, while foreheads and cheeks are both skin colored, they have markedly different shapes.

## 1   Introduction

Many computer vision tasks require the use of appearance and shape models to represent objects in the scene. The choices for appearance models range from histogram-based representations that throws away spatial information, to template-based representations that try to capture the entire spatial layout of the objects but cope poorly with articulation, deformation or variation in appearance. In the middle of this spectrum lie patch-based models that aim to find the right balance between the two extremes.

However, a central problem with existing patch-based models is that there is no way to choose the shape and size of a patch; typically a predefined set of patch sizes and shapes (often rectangles or circles) are used. We believe that natural images can provide enough cues to allow patches to be discovered of varying shape and size corresponding to the shape and size of object parts present in the images. Indeed, we will show that the patches discovered by the jigsaw model can become strongly associated with semantic object parts.

With this motivation, we introduce a generative model for a set of images that learns to extract irregularly shaped and sized patches from a latent image which are combined to generate each training image. We call this latent image a *jigsaw* as it contains all the necessary 'jigsaw pieces' that can be used to generate the target image set. We present an inference algorithm for learning the jigsaw and for finding the jigsaw pieces that make up each image.

As our proposed jigsaw model is a generative model for an image, it can be readily used as a component in many computer vision applications for both image understanding and image synthesis. These include object recognition, detection, image segmentation and image classification, object synthesis, image de-noising, super resolution, texture transfer between images and image in-painting. In fact, the jigsaw model is likely to be useable as a direct replacement for a fixed patch model in any existing patch-based system.

## 2  Related work

The closest work to ours is the epitome model of Jojic et al. [1]. This is a generative model for image patches, or alternatively a model for images if patches that share coordinates in the image are averaged together (although this averaging often leads to a blurry result). Epitomes are learned using a set of fixed shaped patches over a small range of sizes. In contrast, in the jigsaw model, the inference process chooses appropriately shaped and sized pieces from the training images when learning the jigsaw. The difference between these two models is illustrated in section 4.

Our work also closely relates to the seminal work of Freeman et al. [2] that proposed a general machinery for inferring underlying scenes from images, with goals such as in optical flow estimation and super-resolution. They define a Markov random field over image patches and infer the hidden scene representation using belief propagation. Again, they use a set of fixed size image patches, hoping to reach a reasonable trade-off between capturing sufficient statistics in each patch, and disambiguating different kinds of features. Along these lines, Markov random field with larger cliques have also been used to capture the statistic of natural images, such as the field of experts model proposed in [3] which represents the field potentials as non-linear functions of linear filters. Again, the underlying linear filters are applied to patches of a fixed size.

In the domain of image synthesis the work of Freeman et al. [2] has inspired many patch-based synthesis algorithms including super resolution, texture transfer, image in-painting or photo synthesis. They can be viewed as a data-driven way of sampling from the Markov random field with high-order cliques given by the overlapping patches. The texture synthesis and transfer algorithm of Efros et al. [4] constructs a new image by greedily selecting overlapping patches so that the seam transition is not visible. Whilst this work does allow different patch shapes, it does not learn patch appearance since it works from a supplied texture image. Recently a similar approach has been proposed in [5] for synthesising a collage image from a given set of input images, although in this case a probabilistic model is defined and optimised.

Patch-based models are also widely applied in object recognition research [6, 7, 8, 9, 10]. These models use hand-selected patch shapes (typically rectangles) which can lead to poor results given that different object parts have different sizes and shapes. In fact, the use of fixed patches reduces accuracy when the object part is of different size and shape than the chosen patch; in this case, the patch model has to cope with the variability outside the object part. This effect is particularly evident when the part is at the edge of the object as the model then has to try and capture the variability of the background. In addition, such models ignore the shape of the object part which is frequently much more discriminative than appearance alone.

The paper is structured as follows: In section 3 we introduce the probabilistic model and describe a method for performing learning and inference in the model. In section 4 we show results for synthetic and real data and present a comparison to the epitome model. Finally, in section 5, we discuss possible extensions to the model.

## 3  Probabilistic model

This section describes the probabilistic model that we use to learn a jigsaw from a set of training images. We aim to learn a jigsaw such that, given an image set, pieces of the jigsaw image satisfy the following criteria:

- each piece is similar in appearance and shape to several regions of the training images;
- any of the training images can be approximately reconstructed using only pieces from the jigsaw (a piece may be used more than once in a single image);
- pieces are as large as possible for a particular accuracy of reconstruction.

Thus, while allowing the jigsaw pieces to have arbitrary shape, we ensure that such pieces are shared across the entire image set, exhaustively explain the input image set, and are also large enough to be discriminative. By meeting these criteria, we can capture both the appearance and the shape of repeated image structures, for example, eyes, noses and mouths in a set of face images.

We define a jigsaw $J$ to be an image such that each pixel $\mathbf{z}$ in $J$ has an intensity value $\mu(\mathbf{z})$ and an associated variance $\lambda^{-1}(\mathbf{z})$ (so $\lambda$ is the inverse variance, also called the precision). A set of spatially

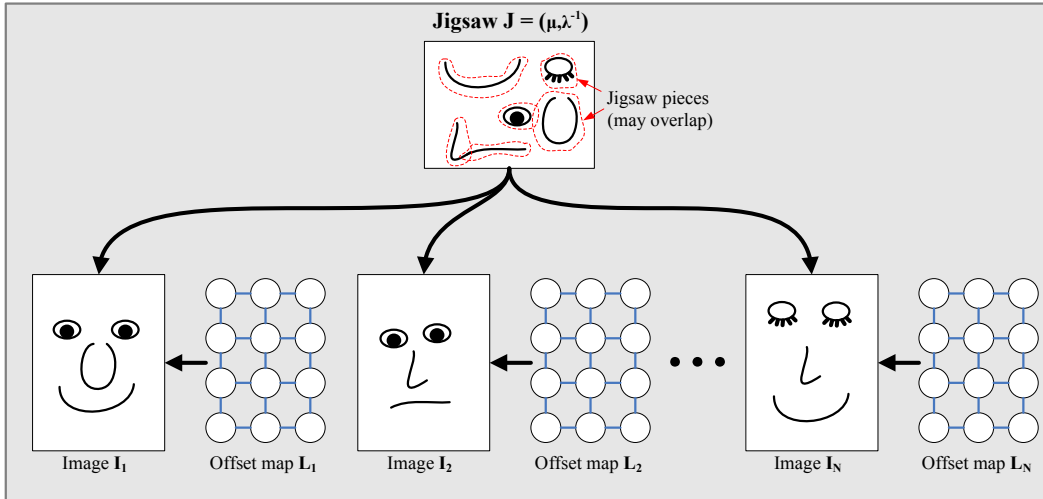

Figure 1: Graphical model showing how the jigsaw $\mathbf{J}$ is used to generate a set of images $\mathbf{I}_1 \ldots \mathbf{I}_N$ by combining the jigsaw pieces in different ways. Each image has a corresponding offset map $\mathbf{L}$ which defines the jigsaw pieces used to generate that image (see text for details). Notice that several jigsaw pieces can overlap and hence share parts of their appearance.

grouped pixels in $J$ is a jigsaw piece. We can combine many of these pieces to generate images, noting that pixels in the jigsaw be re-used in multiple pieces.

Our probabilistic model is a generative image model which generates an image by joining together pieces of the jigsaw and then adding Gaussian noise of variance given by the jigsaw. For each image $\mathbf{I}$, we have an associated *offset map* $\mathbf{L}$ of the same size which determines the jigsaw pieces used to make that image. This offset map defines a position in the jigsaw for each pixel in the image (more than one image pixel can map to the same jigsaw pixel). Each entry in the offset map is a two-dimensional offset $\mathbf{l}_i = (l_x, l_y)$, which maps a 2D point $i$ in the image to a 2D point $\mathbf{z}$ in the jigsaw using $\mathbf{z} = (i - \mathbf{l}_i) \bmod |J|$, where $|J| = (\text{width}, \text{height})$ are the dimensions of the jigsaw. Notice that if two adjacent pixels in the image have the same offset label, then they map to adjacent pixels in the jigsaw. Figure 1 provides a schematic view of the overall probabilistic model, as it is used to generate a set of $N$ face images.

Given this mapping and the jigsaw, the probability distribution of an image is assumed to be independent for each pixel and is given by

$$P(\mathbf{I} \mid \mathbf{J}, \mathbf{L}) = \prod_i \mathcal{N}(I(i); \mu(i - \mathbf{l}_i), \lambda(i - \mathbf{l}_i)^{-1}) \tag{1}$$

where the product is over image pixel positions and both subtractions are modulo $|J|$.

We want the images to consist of coherent pieces of the jigsaw, and so we define a Markov random field on the offset map to encourage neighboring pixels to have the same offsets.

$$P(\mathbf{L}) = \frac{1}{Z} \exp\left[ -\sum_{(i,j) \in E} \psi(\mathbf{l}_i, \mathbf{l}_j) \right] \tag{2}$$

where $E$ is the set of edges in a 4-connected grid. The interaction potential $\psi$ defines a Pott's model on the offsets:

$$\psi(\mathbf{l}_i, \mathbf{l}_j) = \gamma\, \delta(\mathbf{l}_i \neq \mathbf{l}_j) \tag{3}$$

where $\gamma$ is a parameter which influences the typical size of the learned jigsaw pieces. Currently, $\gamma$ is set to give the largest pieces whilst maintaining reasonable quality when the image is reconstructed from the jigsaw.

When learning the jigsaw, it is possible for regions of the jigsaw to be unused, that is, to have no image pixels mapped to them. To allow for this case, we define a Normal-Gamma prior on $\mu$ and $\lambda$

for each jigsaw pixel $\mathbf{z}$,

$$P(\mathbf{J}) = \prod_{\mathbf{z}} \mathcal{N}(\mu(\mathbf{z}); \mu_0, (\beta\lambda(\mathbf{z}))^{-1}) \, \mathrm{Gamma}(\lambda(\mathbf{z}); a, b). \tag{4}$$

This prior means that the behaviour of the model is well defined for unused regions. For our experiments, we fix the hyperparameters $\mu$ to .5, $\beta$ to 1, $b$ to three times the inverse data variance and $a$ to the square of $b$. The local interaction strength $\gamma$ is set to 5 per channel.

**Inference and learning:** The model defines the joint probability distribution on a jigsaw $\mathbf{J}$, a set of images $\mathbf{I}_1 \ldots \mathbf{I}_N$, and their offset maps $\mathbf{L}_1 \ldots \mathbf{L}_N$ to be

$$P\left(\mathbf{J}, \{\mathbf{I}, \mathbf{L}\}_1^N\right) = P(\mathbf{J}) \prod_{n=1}^{N} P(\mathbf{I}_n | \mathbf{J}, \mathbf{L}_n) P(\mathbf{L}). \tag{5}$$

When learning a jigsaw, the image set $\mathbf{I}_1 \ldots \mathbf{I}_N$ is known and we aim to achieve MAP learning of the remaining variables. In other words, our goal is to find the jigsaw $\mathbf{J}$ and offset maps $\mathbf{L}_1 \ldots \mathbf{L}_N$ that maximise the joint probability (5).

We achieve this in an iterative manner. First, the jigsaw is initialised by setting the precisions $\lambda$ to the expected value under the prior $b/a$ and the means $\mu$ to Gaussian noise with the same mean and variance as the data. Given this initialisation, the offset maps are updated for each image by applying the alpha-expansion graph-cut algorithm of [11] (note that our energy is submodular, also known as regular). Whilst this process will not necessarily find the most probable offset map, it is guaranteed to find at least a strong local minimum such that no single expansion move can increase (5).

Given the inferred offset maps, the jigsaw $\mathbf{J}$ that maximises $P\left(\mathbf{J}, \{\mathbf{I}, \mathbf{L}\}_1^N\right)$ can be found analytically. This is achieved for a jigsaw pixel $\mathbf{z}$, the optimal mean $\mu^\star$ and precision $\lambda^\star$ by using

$$\mu^\star = \frac{\beta\mu_0 + \sum_{\mathbf{x} \in X(\mathbf{z})} I(\mathbf{x})}{\beta + |X(\mathbf{z})|} \tag{6}$$

$$\lambda^{-1\star} = \frac{b + \beta\mu_0^2 - (\beta + |X(\mathbf{z})|)(\mu^\star)^2 + \sum_{\mathbf{x} \in X(\mathbf{z})} I(\mathbf{x})^2}{a + |X(\mathbf{z})|} \tag{7}$$

where $\mathbf{X}(\mathbf{z})$ is the set of image pixels that are mapped to the jigsaw pixel $\mathbf{z}$ across all images. We iterate between finding the offset maps holding the jigsaw fixed, and updating the jigsaw using the recently updated offset maps.

When inference has converged, we apply a clustering step to determine the jigsaw pieces (in future we plan to extend the model so that this clustering arises directly during learning). Regions of the image are placed in clusters according to the degree of overlap they have in the jigsaw. The degree of overlap is measured as the ratio of the intersection to the union of the two regions of the jigsaw the image regions map to. This has the effect of clustering image regions by both appearance and shape. Each cluster then corresponds to a region of the jigsaw with an (approximately) consistent shape that explains a large number of image regions.

## 4 Results

**A toy example:** In this experiment, we applied our model to the hand-crafted 150x150 RGB image shown in Fig. 2a. This image was constructed by placing four distinct objects (star, triangle, square and circle), at random positions on a black background image, with the pixels from the more recently placed object replacing the previously drawn pixels. Hence, we can see substantial amount of occlusion of parts of these objects. Using this image as the only input, we would like our model to automatically infer the appearances and shapes of the objects present in the image.

Existing patch-based models are not well-suited to analyzing this image for two reasons: first, there is no clear way to choose the appropriate patch shapes and sizes, and secondly, even if such a choice is known, it is difficult for these existing methods (such as epitome [1]) to learn the shape as they cannot allow for occlusion boundaries without having an explicit occlusion model. For instance, in [1], a separate shape epitome is learned in conjunction with the appearance epitome so that image patches can be explained as a two-layered composition of appearance patches using the shape patch. However, this type of image is difficult to model with a small number of layers due to the large

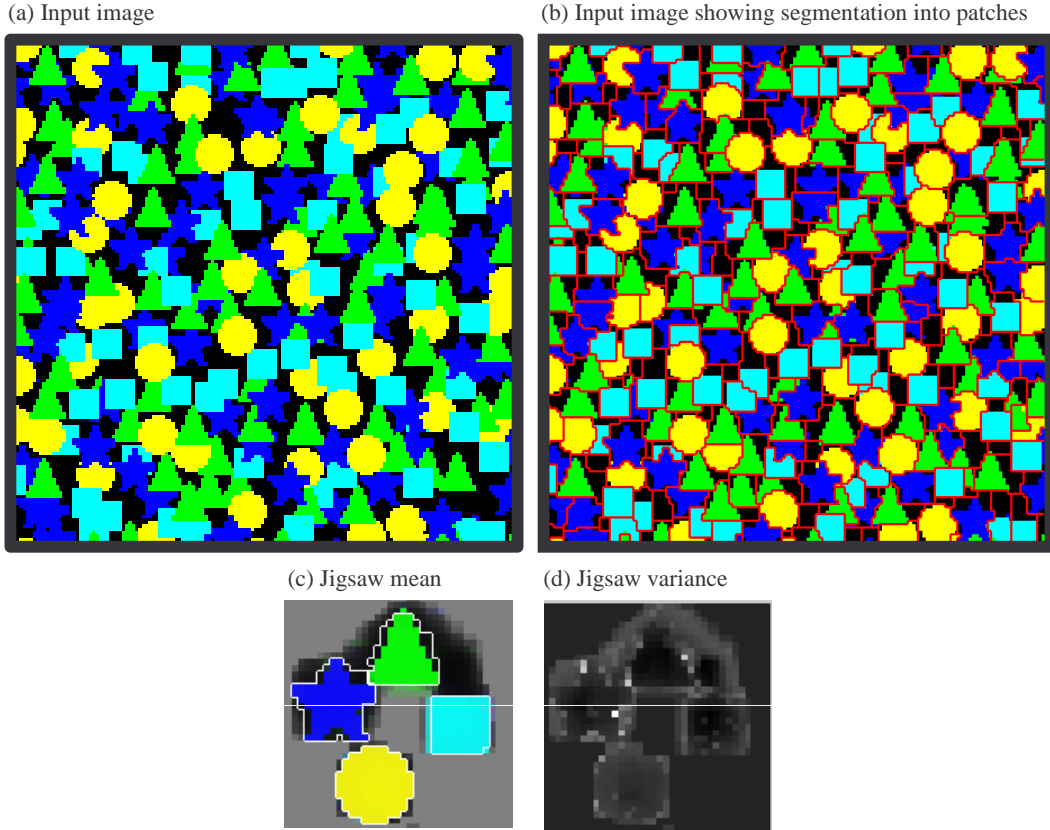

(a) Input image    (b) Input image showing segmentation into patches

(c) Jigsaw mean    (d) Jigsaw variance

Figure 2: Toy example: **(a)** The input image **(b)** Input image with segmentation boundaries super-imposed. Red boundary lines have been drawn on the edge of neighboring pixels that have differing offsets. This segmentation illustrates the different shaped jigsaw pieces found when learning the jig-saw shown in (c)-(d). **(c)** Jigsaw mean with the four most-used jigsaw pieces are outlined in white. **(d)** The jigsaw variance summed across the RGB channels; white is high, black is low.

number of objects present. In contrast, our model can infer any number of overlapping objects, without any explicit modelling of layers or depth. This is because our learning algorithm has the freedom to appropriately adjust a patch's shape and size to explain only a portion of an object without explicitly having to represent a global layer ordering. Moreover, we have the potential to infer the relative depth ordering of neighboring patches by treating rare transitions as occlusions.

Fig. 2b-d shows the results of learning a jigsaw of this toy image. In fig. 2b, we show how the image decomposes into jigsaw pieces. When two neighboring pixels have different labels, they map to non-neighboring locations in the jigsaw. With this understanding, we can look at the change in the labels of the adjacent pixels and plot such a change as a red line. Hence, each region bounded by the red lines indicates a region from the input image being mapped to the jigsaw. From Fig. 2b, we can see that the model has discovered well-defined parts (in this example, objects) present in the image. This is further illustrated in the $36 \times 36$ learned jigsaw whose mean and variance are shown in Fig. 2c,d. The learned jigsaw has captured the shapes and appearances of the four objects and a black region for modelling the background. Under our Bayesian model, pixels in the jigsaw that have never been used in explaining the observation are set to $\mu_0$, which we have fixed to .5 (gray). We can obtain jigsaw pieces by doing the clustering step outlined in Section. 3. In Fig. 2c, we also show the four most-used jigsaw pieces thus obtained by outlining them in white.

**Comparison to epitome model:** In this section, we compare the jigsaw model with the epitome model [1], as applied to the dog image in Fig. 3a. We learned a $32 \times 32$ epitome (Fig. 3d) using all the possible $7 \times 7$ patches from the input image. We then learned a $32 \times 32$ jigsaw (Fig. 3c) such that the average patch area was 49 pixels, the same as in the epitome model. This was achieved by modifying the compatibility parameter $\gamma$. Fig. 3b shows the segmentation of the image after

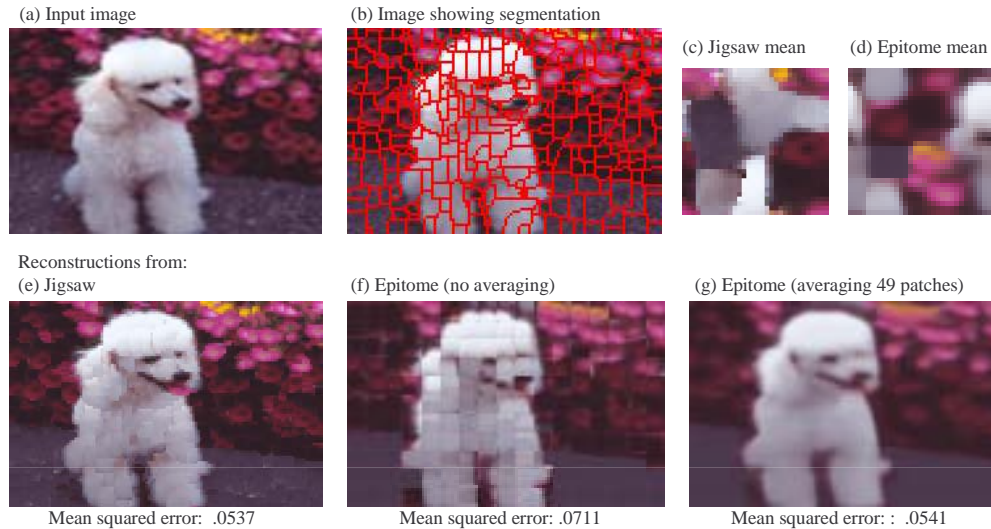

(a) Input image      (b) Image showing segmentation

(c) Jigsaw mean    (d) Epitome mean

Reconstructions from:

(e) Jigsaw      (f) Epitome (no averaging)      (g) Epitome (averaging 49 patches)

Mean squared error: .0537      Mean squared error: .0711      Mean squared error: : .0541

Figure 3: Comparison between jigsaw and epitome. **(a)** The input image **(b)** The segmentation of the image given by the jigsaw model **(c,d)** The means of the learned jigsaw and epitome models **(e)** Reconstruction of the image using the jigsaw **(f)** Reconstruction from the epitome where each image pixel is reconstructed using only one fixed-size patch **(g)** Reconstruction from the epitome where each image pixel is the average of 49 patches. While this reconstruction has similar mean squared error to the jigsaw reconstruction, it is more blurry and less visually pleasing.

learning, with patch boundaries overlaid in red. We can see that the pieces correspond to meaningful regions such as flowers, and also that patch boundaries tend to follow object boundaries.

Comparing Figs. 3c & d, we find that the jigsaw is much less blurred than the epitome and also doesn't have the epitome's artificial 'block' structure. Instead, the boundaries between different textures are placed to allocate the appropriate amount of jigsaw space to each texture, for example, entire flowers are represented as one coherent region. Epitome models can use multi-resolution learning to reduce, but not eliminate, block artifacts. However, whilst this technique can also be applied to jigsaw learning, it has not been found to be necessary in order to obtain a good solution.

In Fig. 3e-g, we compare reconstructions of the input image from the learned jigsaw and epitome models. Since the jigsaw is a generative model for an image, we can reconstruct the image by mapping pixel colors from the jigsaw according to the offset map. When reconstructing from the epitome, we can choose to either use one patch per pixel, or to average a number of patches per pixel. The first approach is most comparable to the jigsaw reconstruction, as it requires only one offset per pixel. However, we find that, as shown in Fig. 3f, the reconstruction is very blocky. When we reconstruct the each pixel from the 49 overlapping patches (Fig. 3g), we find that the reconstruction is overly blurry compared to the jigsaw, despite having a similar mean squared reconstruction error. In addition, this method requires 49 parameters per pixel rather than one and hence is a significantly less compact representation of the image. The reconstruction from the jigsaw is noticeably less blurry and is more visibly pleasing as there is no averaging in the generative process and patch boundaries tend to occur at actual object boundaries.

**Modelling face images:** We next applied the jigsaw model to a set of 100 face images from the Olivetti database at AT&T consisting of 10 different images of 10 people. Each of these grayscale images are of size $64 \times 64$ pixels. We set the jigsaw size to $128 \times 128$ pixels so that the jigsaw has only $1/25$ of the area of the input images combined. Figure 4a shows the inferred segmentation of the images into different shaped and sized pieces (each row contains the images of one person). When the faces depict the same person with similar pose, the resulting segmentations for these images are typically similar, showing that similar jigsaw pieces are being used to explain each image. This can be seen, for instance, from the first row of images in that figure.

Figure 4b shows the mean of the learned jigsaw which can be seen to contain a number of face 'elements' such as eyes, noses etc. To obtain the jigsaw pieces, we applied the clustering step outlined in Section. 3. The obtained clusters are shown in Figure 5(left), which also shows the

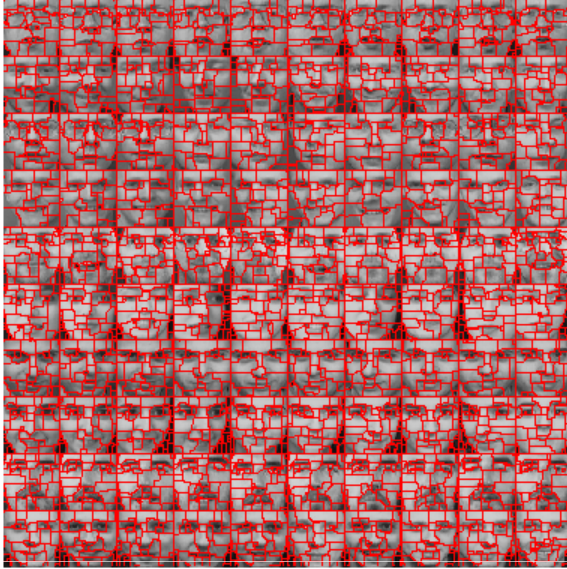

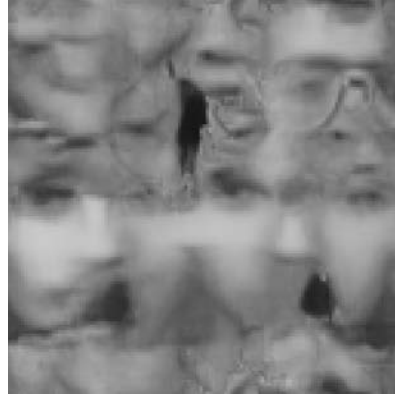

Figure 4: Face images: **(a)** A set of 100 images, each row containing ten different images of the same person, with the segmentation given by the jigsaw model shown in red. **(b)** Jigsaw learned from these face images, see Figure 5 for clustering results.

sharing of these jigsaw pieces. With the jigsaw pieces known, we can now retrieve the regions from the image set that correspond to each jigsaw piece. In Figure 5 (right), we show a random selection of image regions corresponding to several of the most common jigsaw pieces (shown color-coded). What is surprising is that a particular jigsaw piece becomes very strongly associated with a particular face part (far more so than when clustering by appearance alone). Thus, by learning the shape of each jigsaw piece, our model has effectively identified small and large face parts of widely different

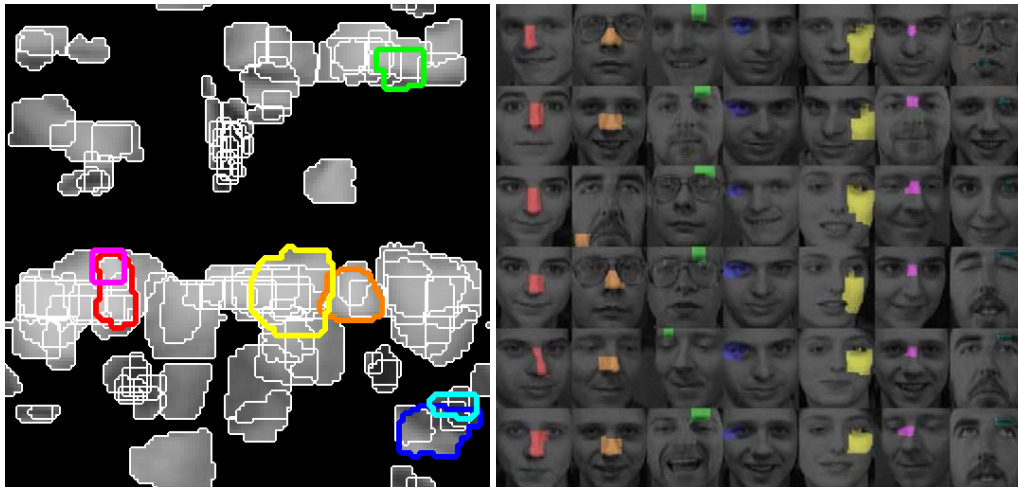

Figure 5: **Left:** The learned face jigsaw of Fig. 4 with overlaid white outlines showing different overlapping jigsaw pieces. For clarity, pieces used five or fewer times are not shown. Areas of the jigsaw not used by the remaining pieces have been blacked out. Seven of the most frequently used jigsaw pieces are shown colored. **Right:** Unsupervised part learning. For each color-coded jigsaw piece in the left image, a column shows *randomly chosen* images from the image set, for which that piece was selected. Notice how these pieces are very strongly associated with different face parts – the model has achieved unsupervised discovery of two different nose shapes, eyes, eyebrows, cheeks etc, despite their widely different shapes and sizes.

shapes and aspect ratios. We can also see from that figure that certain jigsaw pieces are conserved across different people – for example, the nose piece shown in the first column of that figure.

## 5 Discussion

We have presented a generative jigsaw model which is capable of learning the shape, size and appearance of repeated regions in a set of images. We have also shown that, for a set of face images, the learned jigsaw pieces are strongly associated with particular face parts.

Currently, we apply a post-hoc clustering step to learn the jigsaw pieces. This process can be incorporated into the model by extending the pixel offset to include a cluster label and learning the region of jigsaw used by each cluster. We are investigating how best to achieve this.

While we chose a Gaussian as the model for pixel appearance, alternative models can be used, such as histograms, whilst retaining the ability to achieve translation-invariant clustering. Indeed, by using appearance models of other forms, we believe that our model could be used to find repeated structures in other domains such as audio and biology, as well as in images.

Other transformations, such as rotation, scalings and flip, can be incorporated in the model with cost increasing linearly with the number of transformations. We can also extend the model to allow the jigsaw pieces to undergo deformation by favoring neighboring offsets that are similar as well as being identical, using a scheme similar to that of [12].

A practical issue with learning jigsaws is the computational requirement. Every iteration of learning involves solving as many binary graph cuts as there are pixels in the jigsaw. For instance, for the toy example, it took about 30 minutes to learn a $36 \times 36$ jigsaw from a $150 \times 150$ image. We have since developed a significantly faster inference algorithm based on sparse belief propagation. This speed-up allows the model to be applied to larger image sets and to learn larger jigsaws.

Currently, our model does not explicitly account for multiple sources of appearance variability, such as illumination. This means that the same object under different illuminations, for example, will be modelled by different parts of the jigsaw. To account for this, we are investigating factored variants of the jigsaw which separate out different latent causes of appearance variability. Despite this limitation, however, we are already achieving very promising results when using the jigsaw for image synthesis, motion segmentation and object recognition.

**Acknowledgments:** We acknowledge helpful discussions with Nebojsa Jojic, and thank the reviewers for their valuable feedback.

## References

[1] N. Jojic, B. Frey, and A. Kannan. Epitomic analysis of appearance and shape. In *ICCV*, 2003.

[2] W. Freeman, E. Pasztor, and O. Carmichael. Learning low-level vision. *IJCV*, 40(1), 2000.

[3] S. Roth and M. J. Black. Fields of experts: A framework for learning image priors. In *Proceedings of IEEE CVPR*, 2005.

[4] A. Efros and W. Freeman. Image quilting for texture synthesis and transfer. In *ACM Transactions on Graphics (Siggraph)*, 2001.

[5] C. Rother, S. Kumar, V. Kolmogorov, and A. Blake. Digital tapestry. In *Proc. Conf. Computer Vision and Pattern Recognition*, 2005.

[6] R. Fergus, P. Perona, and A. Zisserman. Object class recognition by unsupervised scale-invariant learning. In *CVPR*, volume 2, pages 264–271, June 2003.

[7] B. Leibe and B. Schiele. Interleaved object categorization and segmentation. In *BMVC*, 2003.

[8] E. Borenstein, E. Sharon, and S. Ullman. Combining top-down and bottom-up segmentation. In *Proceedings IEEE workshop on Perceptual Organization in Computer Vision, CVPR 2004*, 2004.

[9] E. Borenstein and S. Ullman. Class-specific, top-down segmentation. In *Proceedings of ECCV*, 2003.

[10] D. Huttenlocher, D. Crandall, and P. Felzenszwalb. Spatial priors for part-based recognition using statistical models. In *Proceedings of IEEE CVPR*, 2005.

[11] Y Boykov, O. Veksler, and R. Zabih. Fast approximate energy minimization via graph cuts. *PAMI*, 23(11), 2001.

[12] J. Winn and J. Shotton. The layout consistent random field for recognizing and segmenting partially occluded objects. In *Proceedings of IEEE CVPR*, 2006.
